# Non-Gaussian Component Analysis: a Semi-parametric Framework for Linear Dimension Reduction

**G. Blanchard**[1], **M. Sugiyama**[1,2], **M. Kawanabe**[1], **V. Spokoiny**[3], **K.-R. Müller**[1,4]

[1] Fraunhofer FIRST.IDA, Kekuléstr. 7, 12489 Berlin, Germany

[2] Dept. of CS, Tokyo Inst. of Tech., 2-12-1, O-okayama, Meguro-ku, Tokyo, 152-8552, Japan

[3] Weierstrass Institute and Humboldt University, Mohrenstr. 39, 10117 Berlin, Germany

[4] Dept. of CS, University of Potsdam, August-Bebel-Strasse 89, 14482 Potsdam, Germany

spokoiny@wias-berlin.de    {blanchar,sugi,nabe,klaus}@first.fhg.de

## Abstract

We propose a new *linear* method for dimension reduction to identify non-Gaussian components in high dimensional data. Our method, NGCA (non-Gaussian component analysis), uses a very general semi-parametric framework. In contrast to existing projection methods we define what is *un*interesting (Gaussian): by projecting out uninterestingness, we can estimate the relevant non-Gaussian subspace. We show that the estimation error of finding the non-Gaussian components tends to zero at a parametric rate. Once NGCA components are identified and extracted, various tasks can be applied in the data analysis process, like data visualization, clustering, denoising or classification. A numerical study demonstrates the usefulness of our method.

## 1 Introduction

Suppose $\{X_i\}_{i=1}^{n}$ are i.i.d. samples in a high dimensional space $\mathbb{R}^d$ drawn from an unknown distribution with density $p(x)$. A general multivariate distribution is typically too complex to analyze from the data, thus dimensionality reduction is necessary to decrease the complexity of the model (see, e.g., [4, 11, 10, 12, 1]). We will follow the rationale that in most real-world applications the 'signal' or 'information' contained in the high-dimensional data is essentially non-Gaussian while the 'rest' can be interpreted as high dimensional Gaussian noise. Thus we implicitly fix what is *not* interesting (Gaussian part) and learn its orthogonal complement, i.e. what is interesting. We call this approach non-Gaussian components analysis (NGCA).

We want to emphasize that we do *not* assume the Gaussian components to be of *smaller* order of magnitude than the signal components. This setting therefore excludes the use of common (nonlinear) dimensionality reduction methods such as Isomap [12], LLE [10], that are based on the assumption that the data lies, say, on a lower dimensional manifold, up to some small noise distortion. In the restricted setting where the number of Gaussian components is *at most one* and all the non-Gaussian components are mutually independent, *Independent Component Analysis (ICA)* techniques (e.g., [9]) are applicable to identify the non-Gaussian subspace.

A framework closer in spirit to NGCA is that of *projection pursuit (PP)* algorithms [5, 7, 9], where the goal is to extract non-Gaussian components in a general setting, i.e., the number of Gaussian components can be more than one and the non-Gaussian components can be dependent. Projection pursuit methods typically proceed by fixing a *single* index which measures the non-Gaussianity (or 'interestingness') of a projection direction. This index is then optimized to find a good direction of projection, and the procedure is iterated to find further directions. Note that some projection indices are suitable for finding super-Gaussian components (heavy-tailed distribution) while others are suited for identifying sub-Gaussian components (light-tailed distribution) [9]. Therefore, traditional PP algorithms may not work effectively if the data contains, say, both super- and sub-Gaussian components.

Technically, the NGCA approach to identify the non-Gaussian subspace uses a very general semi-parametric framework based on a central property: there exists a linear mapping $h \mapsto \beta(h) \in \mathbb{R}^d$ which, to any *arbitrary* (smooth) nonlinear function $h : \mathbb{R}^d \to \mathbb{R}$, associates a vector $\beta$ lying in the non-Gaussian subspace. Using a whole family of different nonlinear functions $h$ then yields a family of different vectors $\widehat{\beta}(h)$ which all approximately lie in, and span, the non-Gaussian subspace. We finally perform PCA on this family of vectors to extract the principal directions and estimate the target space. Our main theoretical contribution in this paper is to prove consistency of the NGCA procedure, i.e. that the above estimation error vanishes at a rate $\sqrt{\log(n)/n}$ with the sample size $n$. In practice, we consider functions of the particular form $h_{\omega,a}(x) = f_a(\langle \omega, x \rangle)$ where $f$ is a function class parameterized, say, by a parameter $a$, and $\|\omega\| = 1$.

Apart from the conceptual point, defining uninterestingness as the point of departure instead of interestingness, another way to look at our method is to say that it allows the combination of information coming from different indices $h$: here the above function $f_a$ (for fixed $a$) plays a role similar to that of a non-Gaussianity index in PP, but we do combine a rich family of such functions (by varying $a$ and even by considering several function classes at the same time). The important point here is while traditional projection pursuit does not provide a well-founded justification for combining directions obtained from different indices, our framework allows to do precisely this – thus implicitly selecting, in a given family of indices, the ones which are the most informative for the data at hand (while always maintaining consistency).

In the following section we will outline our main theoretical contribution, a novel semi-parametric theory for *linear* dimension reduction. Section 3 discusses the algorithmic procedures and simulation results underline the usefulness of NGCA; finally a brief conclusion is given.

## 2   Theoretical framework

**The model.**   We assume the unknown probability density function $p(x)$ of the observations in $\mathbb{R}^d$ is of the form

$$p(x) = g(Tx)\phi_\Gamma(x), \tag{1}$$

where $T$ is an unknown linear mapping from $\mathbb{R}^d$ to another space $\mathbb{R}^m$ with $m \leq d$, $g$ is an unknown function on $\mathbb{R}^m$, and $\phi_\Gamma$ is a centered Gaussian density with unknown covariance matrix $\Gamma$. The above decomposition may be possible for any density $p$ since $g$ can be any function. Therefore, this decomposition is not restrictive in general.

Note that the model (1) includes as particular cases both the pure parametric ($m = 0$) and pure non-parametric ($m = d$) models. We effectively consider an intermediate case where $d$ is large and $m$ is rather small. In what follows we denote by $\mathcal{I}$ the $m$-dimensional *linear* subspace in $\mathbb{R}^d$ generated by the dual operator $T^\top$:

$$\mathcal{I} = Ker(T)^\perp = Range(T^\top).$$

We call $\mathcal{I}$ the *non-Gaussian subspace*. Note how this definition implements the general point of view outlined in the introduction: by this model we define rather what is considered *uninteresting*, i.e. the null space of $T$; the target space is defined indirectly as the orthogonal of the uninteresting component. More precisely, using the orthogonal decomposition $X = X_0 + X_\mathcal{I}$, where $X_0 \in Ker(T)$ and $X_\mathcal{I} \in \mathcal{I}$, equation (1) implies that conditionally to $X_\mathcal{I}$, $X_0$ has a Gaussian distribution. $X_0$ is therefore 'not interesting' and we wish to project it out.

Our goal is therefore to estimate $\mathcal{I}$ by some subspace $\widehat{\mathcal{I}}$ computed from i.i.d. samples $\{X_i\}_{i=1}^n$ which follows the distribution with density $p(x)$. In this paper we assume the effective dimension $m$ to be known or fixed *a priori* by the user. Note that we do *not* estimate $\Gamma$, $g$, and $T$ when estimating $\mathcal{I}$.

**Population analysis.** The main idea underlying our approach is summed up in the following Proposition (proof in Appendix). Whenever variable $X$ has covariance matrix identity, this result allows, from an *arbitrary* smooth real function $h$ on $\mathbb{R}^d$, to find a vector $\beta(h) \in \mathcal{I}$.

**Proposition 1** *Let $X$ be a random variable whose density function $p(x)$ satisfies* (1) *and suppose that $h(x)$ is a smooth real function on $\mathbb{R}^d$. Assume furthermore that $\Sigma = \mathbb{E}\left[XX^\top\right] = I_d$. Then under mild regularity conditions the following vector belongs to the target space $\mathcal{I}$:*

$$\beta(h) = \mathbb{E}\left[\nabla h - Xh(X)\right]. \tag{2}$$

**Estimation using empirical data.** Since the unknown density $p(x)$ is used to define $\beta$ by Eq.(2), one can not directly use this formula in practice, and it must be approximated using the empirical data. We therefore have to estimate the population expectations using empirical ones. A bound on the corresponding approximation error is then given by the following theorem:

**Theorem 1** *Let $h$ be a smooth function. Assume that $\sup_y \max\left(\|\nabla h(y)\|, \|h(y)\|\right) < B$ and that $X$ has covariance matrix $\mathbb{E}\left[XX^\top\right] = I_d$ and is such that for some $\lambda_0 > 0$:*

$$\mathbb{E}\left[\exp\left(\lambda_0 \|X\|\right)\right] \le a_0 < \infty. \tag{3}$$

*Denote $\widetilde{h}(x) = \nabla h(x) - xh(x)$. Suppose $X_1, \ldots, X_n$ are i.i.d. copies of $X$ and define*

$$\widehat{\beta}(h) = \frac{1}{n}\sum_{i=1}^n \widetilde{h}(X_i), \text{ and } \widehat{\sigma}(h) = \frac{1}{n}\sum_{i=1}^n \left\|\widetilde{h}(X_i) - \widehat{\beta}(h)\right\|^2; \tag{4}$$

*then with probability $1 - 4\delta$ the following holds:*

$$dist\left(\widehat{\beta}(h), \mathcal{I}\right) \le 2\sqrt{\widehat{\sigma}(h)\frac{\log \delta^{-1} + \log d}{n}} + C(\lambda_0, a_0, B, d)\left(\frac{\log(n\delta^{-1})\log \delta^{-1}}{n^{\frac{3}{4}}}\right).$$

**Comments.** **1.** The proof of the theorem relies on standard tools using Chernoff's bounding method and is omitted for space. In this theorem, the covariance matrix of $X$ is assumed to be *known* and equal to identity which is not a realistic assumption; in practice, we use a standard "whitening" procedure (see next section) using the empirical covariance matrix. Of course there is an additional error coming from this step, since the covariance matrix is also estimated empirically. In the extended version of the paper [3], we prove (under somewhat stronger assumptions) a bound for the entirely empirical procedure including whitening, resulting in an approximation error of the same order in $n$ (up to a logarithmic factor). This result was omitted here due to space constraints.

**2.** Fixing $\delta$, Theorem 1 implies that the vector $\widehat{\beta}(h)$ obtained from any $h(x)$ converges to the unknown non-Gaussian subspace $\mathcal{I}$ at a "parametric" rate of order $1/\sqrt{n}$. Furthermore, the theorem gives us an estimation of the relative size of the estimation error for different functions $h$ through the (computable from the data) factor $\sqrt{\widehat{\sigma}(h)}$ in the main term of the bound. This suggests using this quantity as a renormalizing factor so that the typical approximation error is (roughly) independent of the function $h$ used. This normalization principle will be used in the main procedure.

**3.** Note the theorem results in an *exponential deviation inequality* (the dependence in the confidence level $\delta$ is logarithmic). As a consequence, using the union bound over a finite net, we can obtain as a corollary of the above theorem a uniform deviation bound of the same form over a (discretized) set of functions (where the log-cardinality of the set appears as an additional factor). For instance, if we consider a $1/n$-discretization net of functions with $d$ parameters, hence of size $\mathcal{O}(n^d)$, then the above bounds holds uniformly when replacing the $\log \delta^{-1}$ term by $d \log n + \log \delta^{-1}$. This does not change fundamentally the bound (up to an additional complexity factor $\sqrt{d \log(n)}$), and justifies that we consider simultaneously such a family of functions in the main algorithm.

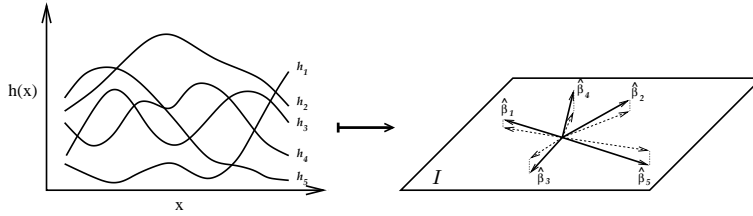

Figure 1: The NGCA main idea: from a varied family of real functions $h$, compute a family of vectors $\widehat{\beta}$ belonging to the target space up to small estimation error.

## 3 The NGCA algorithm

In the last section, we have established that given an arbitrary smooth real function $h$ on $\mathbb{R}^d$, we are able to construct a vector $\widehat{\beta}(h)$ which belongs to the target space $\mathcal{I}$ up to a small estimation error. The main idea is now to consider a large family of such functions $(h_k)$, giving rise to a family of vectors $\widehat{\beta}_k$ (see Fig. 1). Theorem 1 ensures that the estimation error remains controlled uniformly, and we can also normalize the vectors such that the estimation error is of the same order for all vectors (see Comments 2 and 3 above). Under this condition, it can be shown that vectors with a longer norm are more informative about the target subspace, and that vectors with too small a norm are uninformative. We therefore throw out the smaller vectors, then estimate the target space $\mathcal{I}$ by applying a principal components analysis to the remaining vector family.

In the proposed algorithm we will restrict our attention to functions of the form $h_{f,\omega}(x) = f(\langle \omega, x \rangle)$, where $\omega \in \mathbb{R}^d, \|\omega\| = 1$, and $f$ belongs to a finite family $\mathcal{F}$ of smooth real functions of real variable. Our theoretical setting allows to ensure that the approximation error remains small uniformly over $\mathcal{F}$ and $\omega$ (rigorously, $\omega$ should be restricted to a finite $\varepsilon$-net of the unit sphere in order to consider a finite family of functions: in practice we will overlook this weak restriction). However, it is not feasible in practice to sample the whole parameter space for $\omega$ as soon as it has more than a few dimensions. To overcome this difficulty, we advocate using a well-known PP algorithm, FastICA [8], as a proxy to find good candidates for $\omega_f$ for a fixed $f$. Note that this does not make NGCA equivalent to FastICA: the important point is that FastICA, as a stand-alone procedure, requires to fix

the "index function" $f$ beforehand. The crucial novelty of our method is that we provide a theoretical setting and a methodology which allows to *combine* the results of this projection pursuit method when used over a possibly large spectrum of arbitrary index functions $f$.

---

NGCA ALGORITHM.
*Input:* Data points $(X_i) \in \mathbb{R}^d$, dimension $m$ of target subspace.
*Parameters:* Number $T_{\max}$ of FastICA iterations; threshold $\epsilon$;
family of real functions $(f_k)$.

**Whitening.**
  The data $X_i$ is recentered by subtracting the empirical mean.
  Let $\widehat{\Sigma}$ denote the empirical covariance matrix of the data sample $(X_i)$;
  put $\widehat{Y}_i = \widehat{\Sigma}^{-\frac{1}{2}} X_i$ the empirically whitened data.

**Main Procedure.**
  Loop on $k = 1, \dots, L$:
    Draw $\omega_0$ at random on the unit sphere of $\mathbb{R}^d$.
    Loop on $t = 1, \dots, T_{\max}$: *[FastICA loop]*

$$\text{Put } \widehat{\beta}_t \leftarrow \frac{1}{n} \sum_{i=1}^{n} \left( \widehat{Y}_i f_k(\langle \omega_{t-1}, \widehat{Y}_i \rangle) - f'_k(\langle \omega_{t-1}, \widehat{Y}_i \rangle) \omega_{t-1} \right).$$

      Put $\omega_t \leftarrow \widehat{\beta}_t / \|\widehat{\beta}_t\|$.
    End Loop on $t$
    Let $N_i$ be the trace of the empirical covariance matrix of $\widehat{\beta}_{T_{\max}}$:

$$N_i = \frac{1}{n} \sum_{i=1}^{n} \left\| \widehat{Y}_i f_k(\langle \omega_{T_{\max}-1}, \widehat{Y}_i \rangle) - f'_k(\langle \omega_{T_{\max}-1}, \widehat{Y}_i \rangle) \omega_{T_{\max}-1} \right\|^2 - \left\| \widehat{\beta}_{T_{\max}} \right\|^2.$$

    Store $v^{(k)} \leftarrow \widehat{\beta}_{T_{\max}} * \sqrt{n/N_i}$. *[Normalization]*
  End Loop on $k$

**Thresholding.**
  From the family $v^{(k)}$, throw away vectors having norm smaller than threshold $\epsilon$.

**PCA step.**
  Perform PCA on the set of remaining $v^{(k)}$.
  Let $V_m$ be the space spanned by the first $m$ principal directions.

**Pull back in original space.**
  Output: $W_m = \widehat{\Sigma}^{-\frac{1}{2}} V_m$.

---

Summing up, the NGCA algorithm finally consists of the following steps (see above pseudocode): (1) Data whitening (see Comment 1 in the previous section), (2) Apply FastICA to each function $f \in \mathcal{F}$ to find a promising candidate value for $\omega_f$, (3) Compute the corresponding family of vectors $(\widehat{\beta}(h_{f,\omega_f}))_{f \in \mathcal{F}}$ (using Eq. (4)), (4) Normalize the vectors appropriately; threshold and throw out uninformative ones, (5) Apply PCA, (6) Pull back in original space (de-whitening). In the implementation tested, we have used the following forms of the functions $f_k$: $f_\sigma^{(1)}(z) = z^3 \exp(-z^2/2\sigma^2)$ (Gauss-Pow3), $f_b^{(2)}(z) = \tanh(bz)$ (Hyperbolic Tangent), $f_a^{(3)}(z) = \{\sin, \cos\}(az)$ (Fourier). More precisely, we consider discretized ranges for $a \in [0, A], b \in [0, B], \sigma \in [\sigma_{\min}, \sigma_{\max}]$; this gives rise to a finite family $(f_k)$ (which includes *simultaneously* functions of the three different above families).

## 4 Numerical results

**Parameters used.** All the experiments presented where obtained with exactly the same set of parameters: $a \in [0, 4]$ for the Fourier functions; $b \in [0, 5]$ for the Hyperbolic Tangent functions; $\sigma^2 \in [0.5, 5]$ for the Gauss-pow3 functions. Each of these ranges was divided

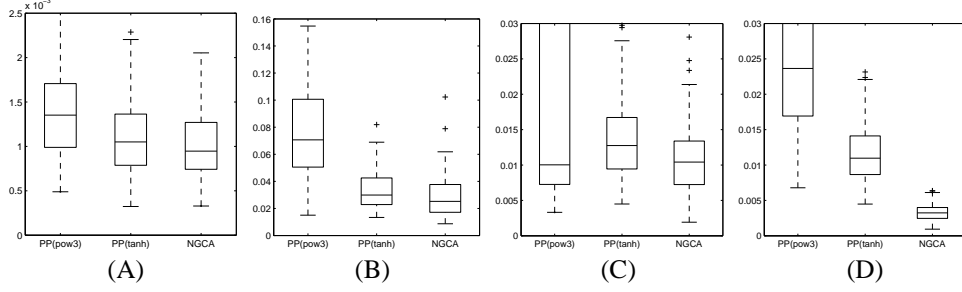

Figure 2: Boxplots of the error criterion $\mathcal{E}(\widehat{\mathcal{I}}, \mathcal{I})$ over 100 training samples of size 1000.

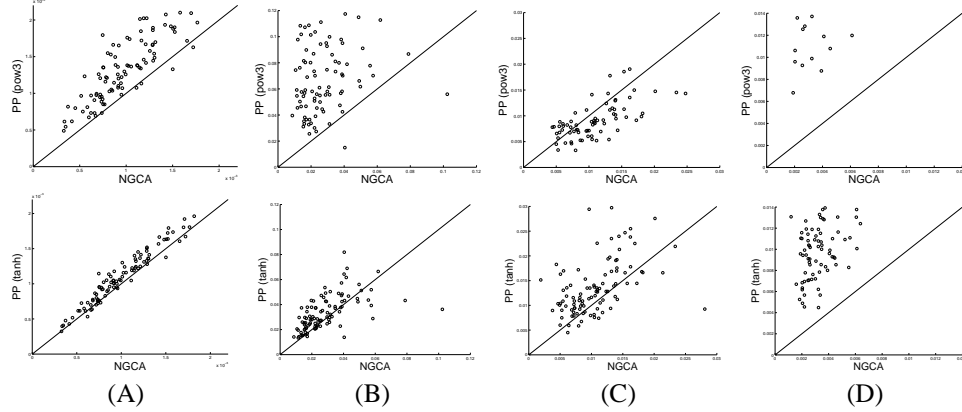

Figure 3: Sample-wise performance comparison plots (for error criterion $\mathcal{E}(\widehat{\mathcal{I}}, \mathcal{I})$) of NGCA versus FastICA; top: versus pow3 index; bottom: versus tanh index. Each point represents a different sample of size 1000. In (C)-top, about 25% of the points corresponding to a failure of FastICA fall outside of the range and were not represented.

into 1000 equispaced values, thus yielding a family $(f_k)$ of size 4000 (Fourier functions count twice because of the sine and cosine parts). Some preliminary calibration suggested to take $\varepsilon = 1.5$ as the threshold under which vectors are not informative. Finally we fixed the number of FastICA iterations $T_{\max} = 10$. With this choice of parameters, with 1000 points of data the computation time is typically of the order of 10 seconds on a modern PC under a Matlab implementation.

**Tests in a controlled setting.** We performed numerical experiments using various synthetic data. We report exemplary results using 4 data sets. Each data set includes 1000 samples in 10 dimensions, and consists of 8-dimensional independent standard Gaussian and 2 non-Gaussian components as follows:
**(A) Simple Gaussian Mixture:** 2-dimensional independent bimodal Gaussian mixtures;
**(B) Dependent super-Gaussian:** 2-dimensional density is proportional to $\exp(-\|x\|)$;
**(C) Dependent sub-Gaussian:** 2-dimensional uniform on the unit circle;
**(D) Dependent super- and sub-Gaussian:** 1-dimensional Laplacian with density proportional to $\exp(-|x_{Lap}|)$ and 1-dimensional dependent uniform $U(c, c+1)$, where $c = 0$ for $|x_{Lap}| \leq \log 2$ and $c = -1$ otherwise.
We compare the NGCA method against stand-alone FastICA with two different index functions. Figure 2 shows boxplots and Figure 3 sample-wise comparison plots, over 100 samples, of the error criterion $\mathcal{E}(\widehat{\mathcal{I}}, \mathcal{I}) = m^{-1} \sum_{i=1}^{m} \|(I_d - P_{\mathcal{I}})\widehat{v}_i\|^2$, where $\{\widehat{v}_i\}_{i=1}^{m}$ is an

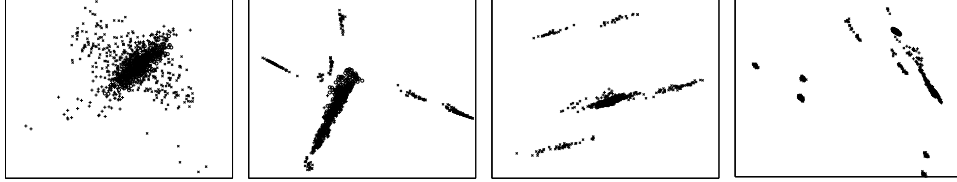

Figure 4: 2D projection of the "oil flow" (12-dimensional) data obtained by different algorithms, from left two right: PCA, Isomap, FastICA (tanh index), NGCA. In each case, the data was first projected in 3D using the respective methods, from which a 2D projection was chosen visually so as to yield the clearest cluster structure. Available label information was not used to determine the projections.

orthonormal basis of $\widehat{\mathcal{I}}$, $I_d$ is the identity matrix, and $P_{\mathcal{I}}$ denotes the orthogonal projection on $\mathcal{I}$. In datasets (A),(B),(C), NGCA appears to be on par with the best FastICA method. As expected, the best index for FastICA is data-dependent: the 'tanh' index is more suited to the super-Gaussian data (B), while the 'pow3' index works best with the sub-Gaussian data (C) (although, in this case, FastICA with this index has a tendency to get caught in local minima, leading to a disastrous result for about 25% of the samples. Note that NGCA does *not* suffer from this problem). Finally, the advantage of the implicit index adaptation feature of NGCA can be clearly observed in the data set (D), which includes both sub- and super-Gaussian components. In this case, neither of the two FastICA index functions taken alone does well, and NGCA gives significantly lower error than either FastICA flavor.

**Example of application for realistic data: visualization and clustering** We now give an example of application of NGCA to visualization and clustering of realistic data. We consider here "oil flow" data, which has been obtained by numerical simulation of a complex physical model. This data was already used before for testing techniques of dimension reduction [2]. The data is 12-dimensional and our goal is to visualize the data, and possibly exhibit a clustered structure. We compared results obtained with the NGCA methodology, regular PCA, FastICA with tanh index and Isomap. The results are shown on Figure 4. A 3D projection of the data was first computed using these methods, which was in turn projected in 2D to draw the figure; this last projection was chosen manually so as to make the cluster structure as visible as possible in each case. The NGCA result appears better with a clearer clustered structure appearing. This structure is only partly visible in the Isomap result; the NGCA method additionally has the advantage of a clear geometrical interpretation (linear orthogonal projection). Finally, datapoints in this dataset are distributed in 3 classes. This information was not used in the different procedures, but we can see *a posteriori* that only NGCA clearly separates the classes in distinct clusters. Clustering applications on other benchmark datasets is presented in the extended paper [3].

## 5 Conclusion

We proposed a new semi-parametric framework for constructing a linear projection to separate an uninteresting, possibly of large amplitude multivariate Gaussian 'noise' subspace from the 'signal-of-interest' subspace. We provide generic consistency results on how well the non-Gaussian directions can be identified (Theorem 1). Once the low-dimensional 'signal' part is extracted, we can use it for a variety of applications such as data visualization, clustering, denoising or classification.
Numerically we found comparable or superior performance to, e.g., FastICA in deflation mode as a generic representative of the family of PP algorithms. Note that in general, PP methods need to pre-specify a projection index with which they search non-Gaussian

components. By contrast, an important advantage of our method is that we are able to simultaneously use several families of nonlinear functions; moreover, also inside a same function family we are able to use an entire range of parameters (such as frequency for Fourier functions). Thus, NGCA provides higher flexibility, and less restricting assumptions *a priori* on the data. In a sense, the functional indices that are the most relevant for the data at hand are automatically selected.

Future research will adapt the theory to simultaneously estimate the dimension of the non-Gaussian subspace. Extending the proposed framework to non-linear projection scenarios [4, 11, 10, 12, 1, 6] and to finding the most discriminative directions using labels are examples for which the current theory could be taken as a basis.

**Acknowledgements:** This work was supported in part by the PASCAL Network of Excellence (EU # 506778).

**Proof of Proposition 1**   Put $\alpha = \mathbb{E}\left[Xh(X)\right]$ and $\psi(x) = h(x) - \alpha^\top x$. Note that $\nabla \psi = \nabla h - \alpha$, hence $\beta(h) = \mathbb{E}\left[\nabla \psi(X)\right]$. Furthermore, it holds by change of variable that

$$\int \psi(x+u)p(x)dx = \int \psi(x)p(x-u)dx.$$

Under mild regularity conditions on $p(x)$ and $h(x)$, differentiating this with respect to $u$ gives

$$\mathbb{E}\left[\nabla \psi(X)\right] = \int \nabla \psi(x)p(x)dx = -\int \psi(x)\nabla p(x)dx = -\mathbb{E}\left[\psi(X)\nabla \log p(X)\right],$$

where we have used $\nabla p(x) = \nabla \log p(x)\, p(x)$. Eq.(1) now implies $\nabla \log p(x) = \nabla \log g(Tx) - \Gamma^{-1}x$, hence

$$\beta(\psi) = -\mathbb{E}\left[\psi(X)\nabla \log g(TX)\right] + \mathbb{E}\left[\psi(X)\Gamma^{-1}X\right]$$
$$= -T^\top \mathbb{E}\left[\psi(X)\nabla g(TX)/g(TX)\right] + \Gamma^{-1}\mathbb{E}\left[Xh(X) - XX^\top \mathbb{E}\left[Xh(X)\right]\right].$$

The last term above vanishes because we assumed $\mathbb{E}\left[XX^\top\right] = I_d$. The first term belongs to $\mathcal{I}$ by definition. This concludes the proof. □

# References

[1] M. Belkin and P. Niyogi. Laplacian eigenmaps for dimensionality reduction and data representation. *Neural Computation*, 15(6):1373–1396, 2003.

[2] C.M. Bishop, M. Svensen and C.K.I. Wiliams. GTM: The generative topographic mapping. *Neural Computation*, 10(1):215–234, 1998.

[3] G. Blanchard, M. Sugiyama, M. Kawanabe, V. Spokoiny, K.-R. Müller. *In search of non-Gaussian components of a high-dimensional distribution*. Technical report of the Weierstrass Institute for Applied Analysis and Stochastics, 2006.

[4] T.F. Cox and M.A.A. Cox. *Multidimensional Scaling*. Chapman & Hall, London, 2001.

[5] J.H. Friedman and J.W. Tukey. A projection pursuit algorithm for exploratory data analysis. *IEEE Transactions on Computers*, 23(9):881–890, 1975.

[6] S. Harmeling, A. Ziehe, M. Kawanabe and K.-R. Müller. Kernel-based nonlinear blind source separation. *Neural Computation*, 15(5):1089–1124, 2003.

[7] P.J. Huber. Projection pursuit. *The Annals of Statistics*, 13:435–475, 1985.

[8] A. Hyvärinen. Fast and robust fixed-point algorithms for independent component analysis. *IEEE Transactions on Neural Networks*, 10(3):626–634, 1999.

[9] A. Hyvärinen, J. Karhunen and E. Oja. *Independent component analysis*. Wiley, 2001.

[10] S. Roweis and L. Saul. Nonlinear dimensionality reduction by locally linear embedding. *Science*, 290(5500):2323–2326, 2000.

[11] B. Schölkopf, A.J. Smola and K.–R. Müller. Nonlinear component analysis as a kernel Eigenvalue problem. *Neural Computation*, 10(5):1299–1319, 1998.

[12] J.B. Tenenbaum, V. de Silva and J.C. Langford. A global geometric framework for nonlinear dimensionality reduction. *Science*, 290(5500):2319–2323, 2000.
